# Variable margin losses for classifier design

**Hamed Masnadi-Shirazi**
Statistical Visual Computing Laboratory,
University of California, San Diego
La Jolla, CA 92039
hmasnadi@ucsd.edu

**Nuno Vasconcelos**
Statistical Visual Computing Laboratory,
University of California, San Diego
La Jolla, CA 92039
nuno@ucsd.edu

## Abstract

The problem of controlling the margin of a classifier is studied. A detailed analytical study is presented on how properties of the classification risk, such as its optimal link and minimum risk functions, are related to the shape of the loss, and its margin enforcing properties. It is shown that for a class of risks, denoted canonical risks, asymptotic Bayes consistency is compatible with simple analytical relationships between these functions. These enable a precise characterization of the loss for a popular class of link functions. It is shown that, when the risk is in canonical form and the link is inverse sigmoidal, the margin properties of the loss are determined by a single parameter. Novel families of Bayes consistent loss functions, of variable margin, are derived. These families are then used to design boosting style algorithms with explicit control of the classification margin. The new algorithms generalize well established approaches, such as LogitBoost. Experimental results show that the proposed variable margin losses outperform the fixed margin counterparts used by existing algorithms. Finally, it is shown that best performance can be achieved by cross-validating the margin parameter.

## 1 Introduction

Optimal classifiers minimize the expected value of a loss function, or risk. Losses commonly used in machine learning are upper-bounds on the zero-one classification loss of classical Bayes decision theory. When the resulting classifier converges asymptotically to the Bayes decision rule, as training samples increase, the loss is said to be Bayes consistent. Examples of such losses include the hinge loss, used in SVM design, the exponential loss, used by boosting algorithms such as AdaBoost, or the logistic loss, used in both classical logistic regression and more recent methods, such as LogitBoost. Unlike the zero-one loss, these losses assign a penalty to examples correctly classified but close to the boundary. This guarantees a classification margin, and improved generalization when learning from finite datasets [1]. Although the connections between large-margin classification and classical decision theory have been known since [2], the set of Bayes consistent large-margin losses has remained small. Most recently, the design of such losses has been studied in [3]. By establishing connections to the classical literature in probability elicitation [4], this work introduced a generic framework for the derivation of Bayes consistent losses. The main idea is that there are three quantities that matter in risk minimization: the loss function $\phi$, a corresponding optimal link function $f_\phi^*$, which maps posterior class probabilities to classifier predictions, and a minimum risk $C_\phi^*$, associated with the optimal link.

While the standard approach to classifier design is to define a loss $\phi$, and then optimize it to obtain $f_\phi^*$ and $C_\phi^*$, [3] showed that there is an alternative: to specify $f_\phi^*$ and $C_\phi^*$, and analytically derive the loss $\phi$. The advantage is that this makes it possible to manipulate the properties of the loss, while *guaranteeing* that it is Bayes consistent. The practical relevance of this approach is illustrated in [3], where a Bayes consistent *robust* loss is derived, for application in problems involving outliers. This

loss is then used to design a robust boosting algorithm, denoted SavageBoost. SavageBoost has been, more recently, shown to outperform most other boosting algorithms in computer vision problems, where outliers are prevalent [5]. The main limitation of the framework of [3] is that it is not totally constructive. It turns out that many pairs $(C_\phi^*, f_\phi^*)$ are compatible with any Bayes consistent loss $\phi$. Furthermore, while there is a closed form relationship between $\phi$ and $(C_\phi^*, f_\phi^*)$, this relationship is far from simple. This makes it difficult to understand how the properties of the loss are influenced by the properties of either $C_\phi^*$ or $f_\phi^*$. In practice, the design has to resort to trial and error, by 1) testing combinations of the latter and, 2) verifying whether the loss has the desired properties. This is feasible when the goal is to enforce a broad loss property, e.g. that a robust loss should be bounded for negative margins [3], but impractical when the goal is to exercise a finer degree of control.

In this work, we consider one such problem: how to control the size of the margin enforced by the loss. We start by showing that, while many pairs $(C_\phi^*, f_\phi^*)$ are compatible with a given $\phi$, one of these pairs establishes a very tight connection between the optimal link and the minimum risk: that $f_\phi^*$ is the derivative of $C_\phi^*$. We refer to the risk function associated with such a pair as a *canonical risk*, and show that it leads to an equally tight connection between the pair $(C_\phi^*, f_\phi^*)$ and the loss $\phi$. For a canonical risk, all three functions can be obtained from each other with one-to-one mappings of trivial analytical tractability. This enables a detailed analytical study of how $C_\phi^*$ or $f_\phi^*$ affect $\phi$. We consider the case where the inverse of $f_\phi^*$ is a sigmoidal function, i.e. $f_\phi^*$ is *inverse-sigmoidal*, and show that this strongly constrains the loss. Namely, the latter becomes 1) convex, 2) monotonically decreasing, 3) linear for large negative margins, and 4) constant for large positive margins. This implies that, for a canonical risk, the choice of a particular link in the inverse-sigmoidal family *only* impacts the behavior of $\phi$ around the origin, i.e. the size of the margin enforced by the loss. This quantity is then shown to depend only on the slope of the sigmoidal inverse-link at the origin. Since this property can be controlled by a single parameter, the latter becomes a margin-tunning parameter, i.e. a parameter that determines the margin of the optimal classifier. This is exploited to design parametric families of loss functions that allow *explicit control* of the classification margin. These losses are applied to the design of novel boosting algorithms of tunable margin. Finally, it is shown that the requirements of 1) a canonical risk, and 2) an inverse-sigmoidal link are not unduly restrictive for classifier design. In fact, approaches like logistic regression or LogitBoost are special cases of the proposed framework. A number of experiments are conducted to study the effect of margin-control on the classification accuracy. It is shown that the proposed variable-margin losses outperform the fixed-margin counterparts used by existing algorithms. Finally, it is shown that cross-validation of the margin parameter leads to classifiers with the best performance on all datasets tested.

## 2 Loss functions for classification

We start by briefly reviewing the theory of Bayes consistent classifier design. See [2, 6, 7, 3] for further details. A classifier $h$ maps a feature vector $\mathbf{x} \in \mathcal{X}$ to a class label $y \in \{-1, 1\}$. This mapping can be written as $h(\mathbf{x}) = sign[p(\mathbf{x})]$ for some function $p : \mathcal{X} \to \mathbb{R}$, which is denoted as the classifier predictor. Feature vectors and class labels are drawn from probability distributions $P_{\mathbf{X}}(\mathbf{x})$ and $P_Y(y)$ respectively. Given a non-negative loss function $L(\mathbf{x}, y)$, the classifier is optimal if it minimizes the risk $R(f) = E_{\mathbf{X}, Y}[L(h(\mathbf{x}), y)]$. This is equivalent to minimizing the conditional risk $E_{Y|\mathbf{X}}[L(h(\mathbf{x}), y)|\mathbf{X} = \mathbf{x}]$ for all $\mathbf{x} \in \mathcal{X}$. It is useful to express $p(\mathbf{x})$ as a composition of two functions, $p(\mathbf{x}) = f(\eta(\mathbf{x}))$, where $\eta(\mathbf{x}) = P_{Y|\mathbf{X}}(1|\mathbf{x})$, and $f : [0, 1] \to \mathbb{R}$ is a *link function*. Classifiers are frequently designed to be optimal with respect to the zero-one loss

$$L_{0/1}(f, y) = \frac{1 - sign(yf)}{2} = \begin{cases} 0, & \text{if } y = sign(f); \\ 1, & \text{if } y \neq sign(f), \end{cases} \tag{1}$$

where we omit the dependence on $\mathbf{x}$ for notational simplicity. The associated conditional risk is

$$C_{0/1}(\eta, f) = \eta \frac{1 - sign(f)}{2} + (1 - \eta)\frac{1 + sign(f)}{2} = \begin{cases} 1 - \eta, & \text{if } f \geq 0; \\ \eta, & \text{if } f < 0. \end{cases} \tag{2}$$

The risk is minimized if

$$\begin{cases} f(\mathbf{x}) > 0 & \text{if } \eta(\mathbf{x}) > \frac{1}{2} \\ f(\mathbf{x}) = 0 & \text{if } \eta(\mathbf{x}) = \frac{1}{2} \\ f(\mathbf{x}) < 0 & \text{if } \eta(\mathbf{x}) < \frac{1}{2} \end{cases} \tag{3}$$

Table 1: Loss $\phi$, optimal link $f_\phi^*(\eta)$, optimal inverse link $[f_\phi^*]^{-1}(v)$ , and minimum conditional risk $C_\phi^*(\eta)$ for popular learning algorithms.

| Algorithm | $\phi(v)$ | $f_\phi^*(\eta)$ | $[f_\phi^*]^{-1}(v)$ | $C_\phi^*(\eta)$ |
|---|---|---|---|---|
| SVM | $\max(1-v,0)$ | $sign(2\eta-1)$ | NA | $1-\|2\eta-1\|$ |
| Boosting | $\exp(-v)$ | $\frac{1}{2}\log\frac{\eta}{1-\eta}$ | $\frac{e^{2v}}{1+e^{2v}}$ | $2\sqrt{\eta(1-\eta)}$ |
| Logistic Regression | $\log(1+e^{-v})$ | $\log\frac{\eta}{1-\eta}$ | $\frac{e^v}{1+e^v}$ | $-\eta\log\eta-(1-\eta)\log(1-\eta)$ |

Examples of optimal link functions include $f^* = 2\eta - 1$ and $f^* = \log\frac{\eta}{1-\eta}$. The associated optimal classifier $h^* = sign[f^*]$ is the well known Bayes decision rule (BDR), and the associated minimum conditional (zero-one) risk is

$$C_{0/1}^*(\eta) = \eta\left(\frac{1}{2} - \frac{1}{2}sign(2\eta-1)\right) + (1-\eta)\left(\frac{1}{2} + \frac{1}{2}sign(2\eta-1)\right). \tag{4}$$

A loss which is minimized by the BDR is Bayes consistent. A number of Bayes consistent alternatives to the 0-1 loss are commonly used. These include the exponential loss of boosting, the log loss of logistic regression, and the hinge loss of SVMs. They have the form $L_\phi(f,y) = \phi(yf)$, for different functions $\phi$. These functions assign a non-zero penalty to small positive $yf$, encouraging the creation of a margin, a property not shared by the 0-1 loss. The resulting *large-margin* classifiers have better generalization than those produced by the latter [1]. The associated conditional risk

$$C_\phi(\eta, f) = \eta\phi(f) + (1-\eta)\phi(-f). \tag{5}$$

is minimized by the link

$$f_\phi^*(\eta) = \arg\min_f C_\phi(\eta, f) \tag{6}$$

leading to the minimum conditional risk function $C_\phi^*(\eta) = C_\phi(\eta, f_\phi^*)$. Table 1 lists the loss, optimal link, and minimum risk of some of the most popular classifier design methods.

Conditional risk minimization is closely related to classical probability elicitation in statistics [4]. Here, the goal is to find the probability estimator $\hat{\eta}$ that maximizes the expected reward

$$I(\eta, \hat{\eta}) = \eta I_1(\hat{\eta}) + (1-\eta)I_{-1}(\hat{\eta}), \tag{7}$$

where $I_1(\hat{\eta})$ is the reward for prediction $\hat{\eta}$ when event $y = 1$ holds and $I_{-1}(\hat{\eta})$ the corresponding reward when $y = -1$. The functions $I_1(\cdot), I_{-1}(\cdot)$ should be such that the expected reward is maximal when $\hat{\eta} = \eta$, i.e.

$$I(\eta, \hat{\eta}) \leq I(\eta, \eta) = J(\eta), \ \ \forall\eta \tag{8}$$

with equality if and only if $\hat{\eta} = \eta$. The conditions under which this holds are as follows.

**Theorem 1.** *[4] Let $I(\eta, \hat{\eta})$ and $J(\eta)$ be as defined in (7) and (8). Then 1) $J(\eta)$ is convex and 2) (8) holds if and only if*

$$I_1(\eta) = J(\eta) + (1-\eta)J'(\eta) \tag{9}$$
$$I_{-1}(\eta) = J(\eta) - \eta J'(\eta). \tag{10}$$

Hence, starting from any convex $J(\eta)$, it is possible to derive $I_1(\cdot), I_{-1}(\cdot)$ so that (8) holds. This enables the following connection to risk minimization.

**Theorem 2.** *[3] Let $J(\eta)$ be as defined in (8) and $f$ a continuous function. If the following properties hold*

1. *$J(\eta) = J(1-\eta)$,*

2. *$f$ is invertible with symmetry*

$$f^{-1}(-v) = 1 - f^{-1}(v), \tag{11}$$

*then the functions $I_1(\cdot)$ and $I_{-1}(\cdot)$ derived with (9) and (10) satisfy the following equalities*

$$
\begin{align}
I_1(\eta) &= -\phi(f(\eta)) \tag{12} \\
I_{-1}(\eta) &= -\phi(-f(\eta)), \tag{13}
\end{align}
$$

*with*

$$\phi(v) = -J[f^{-1}(v)] - (1 - f^{-1}(v))J'[f^{-1}(v)]. \tag{14}$$

Under the conditions of the theorem, $I(\eta, \hat{\eta}) = -C_\phi(\eta, f)$. This establishes a new path for classifier design [3]. Rather than specifying a loss $\phi$ and minimizing $C_\phi(\eta, f)$, so as to obtain whatever optimal link $f_\phi^*$ and minimum expected risk $C_\phi^*(\eta)$ results, it is possible to specify $f_\phi^*$ and $C_\phi^*(\eta)$ and derive, from (14) with $J(\eta) = -C_\phi^*(\eta)$, the underlying loss $\phi$. The main advantage is the ability to control directly the quantities that matter for classification, namely the predictor and risk of the optimal classifier. The only conditions are that $C_\phi^*(\eta) = C_\phi^*(1 - \eta)$ and (11) holds for $f_\phi^*$.

## 3   Canonical risk minimization

In general, given $J(\eta) = -C_\phi^*(\eta)$, there are multiple pairs $(\phi, f_\phi^*)$ that satisfy (14). Hence, specification of either the minimum risk or optimal link does not completely characterize the loss. This makes it difficult to control some important properties of the latter, such as the margin. In this work, we consider an important special case, where such control is possible. We start with a lemma that relates the symmetry conditions, on $J(\eta)$ and $f_\phi^*(\eta)$, of Theorem 2.

**Lemma 3.** *Let $J(\eta)$ be a strictly convex and differentiable function such that $J(\eta) = J(1 - \eta)$. Then $J'(\eta)$ is invertible and*

$$[J']^{-1}(-v) = 1 - [J']^{-1}(v). \tag{15}$$

Hence, under the conditions of Theorem 2, the derivative of $J(\eta)$ has the *same* symmetry as $f_\phi^*(\eta)$. Since this symmetry is the only constraint on $f_\phi^*$, the former can be used as the latter. Whenever this holds, the risk is said to be in canonical form, and $(f^*, J)$ are denoted a canonical pair [6] .

**Definition 1.** *Let $J(\eta)$ be as defined in (8), and $C_\phi^*(\eta) = -J(\eta)$ a minimum risk. If the optimal link associated with $C_\phi^*(\eta)$ is*

$$f_\phi^*(\eta) = J'(\eta) \tag{16}$$

*the risk $C_\phi(\eta, f)$ is said to be in canonical form. $f_\phi^*(\eta)$ is denoted a canonical link and $\phi(v)$, the loss given by (14), a canonical loss.*

Note that (16) does not hold for all risks. For example, the risk of boosting is derived from the convex, differentiable, and symmetric $J(\eta) = -2\sqrt{\eta(1 - \eta)}$. Since this has derivative

$$J'(\eta) = \frac{2\eta - 1}{\sqrt{\eta(1 - \eta)}} \neq \frac{1}{2} \log \frac{\eta}{1 - \eta} = f_\phi^*(\eta), \tag{17}$$

the risk is not in canonical form. What follows from (16) is that *it is possible* to derive a canonical risk for *any* maximal reward $J(\eta)$, including that of boosting ($J(\eta) = -2\sqrt{\eta(1 - \eta)}$). This is discussed in detail in Section 5.

While canonical risks can be easily designed by specifying either $J(\eta)$ or $f_\phi^*(\eta)$, and then using (14) and (16), it is much less clear how to directly specify a loss $\phi(v)$ for which (14) holds with a canonical pair $(f^*, J)$. The following result solves this problem.

**Theorem 4.** *Let $C_\phi(\eta, f)$ be the canonical risk derived from a convex and symmetric $J(\eta)$. Then*

$$\phi'(v) = -[J']^{-1}(-v) = [f_\phi^*]^{-1}(v) - 1. \tag{18}$$

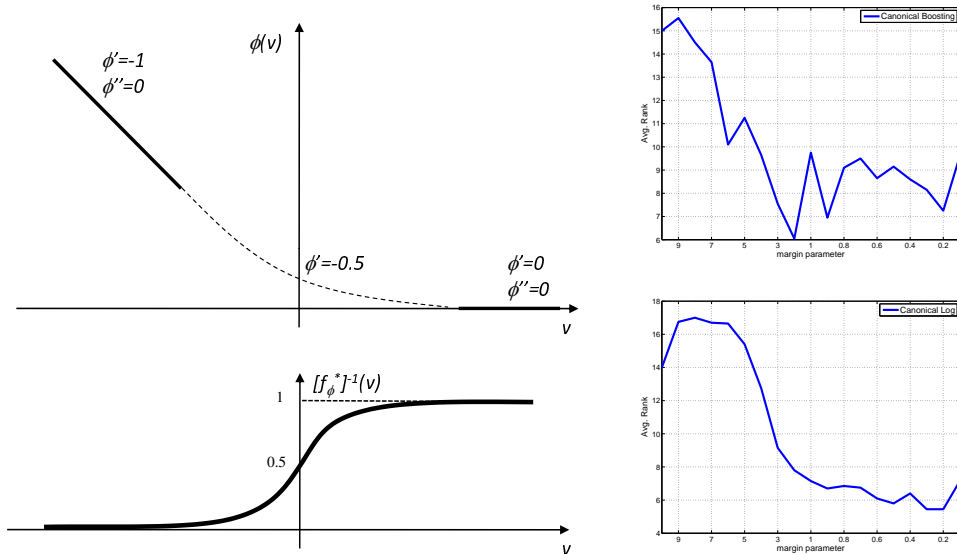

Figure 1: Left: canonical losses compatible with an IS optimal link. Right: Average classification rank as a function of margin parameter, on the UCI data.

This theorem has various interesting consequences. First, it establishes an easy-to-verify necessary condition for the canonical form. For example, logistic regression has $[f_\phi^*]^{-1}(v) = \frac{1}{1+e^{-v}}$ and $\phi'(v) = -\frac{e^{-v}}{1+e^{-v}} = [f_\phi^*]^{-1}(v) - 1$, while boosting has $[f_\phi^*]^{-1}(v) = \frac{1}{1+e^{-2v}}$ and $\phi'(v) = -e^{-v} \neq [f_\phi^*]^{-1}(v) - 1$. This (plus the symmetry of $J$ and $f_\phi^*$) shows that the former is in canonical form but the latter is not. Second, it makes it clear that, up to additive constants, the three components $(\phi, C_\phi^*, \text{ and } f_\phi^*)$ of a canonical risk are related by one-to-one relationships. Hence, it is possible to control the properties of the *three* components of the risk by manipulating a *single* function (which can be any of the three). Finally, it enables a very detailed characterization of the losses compatible with most optimal links of Table 1.

## 4    Inverse-sigmoidal links

Inspection of Table 1 suggests that the classifiers produced by boosting, logistic regression, and variants have sigmoidal inverse links $[f_\phi^*]^{-1}$. Due to this, we refer to the links $f_\phi^*$ as *inverse-sigmoidal* (IS). When this is the case, (18) provides a very detailed characterization of the loss $\phi$. In particular, it can be trivially shown that, letting $f^{(n)}$ be the $n^{th}$ order derivative of $f$, that the following hold

$$\lim_{v \to -\infty} [f_\phi^*]^{-1}(v) = 0 \quad \Leftrightarrow \quad \lim_{v \to -\infty} \phi^{(1)}(v) = -1 \tag{19}$$

$$\lim_{v \to \infty} [f_\phi^*]^{-1}(v) = 1 \quad \Leftrightarrow \quad \lim_{v \to \infty} \phi^{(1)}(v) = 0 \tag{20}$$

$$\lim_{v \to \pm\infty} ([f_\phi^*]^{-1})^{(n)}(v) = 0, n \geq 1 \quad \Leftrightarrow \quad \lim_{v \to \pm\infty} \phi^{(n+1)}(v) = 0, n \geq 1 \tag{21}$$

$$[f_\phi^*]^{-1}(v) \in (0,1) \quad \Leftrightarrow \quad \phi(v) \text{ monotonically decreasing} \tag{22}$$

$$[f_\phi^*]^{-1}(v) \text{ monotonically increasing} \quad \Leftrightarrow \quad \phi(v) \text{ convex} \tag{23}$$

$$[f_\phi^*]^{-1}(0) = .5 \quad \Leftrightarrow \quad \phi^{(1)}(0) = -.5. \tag{24}$$

It follows that, as illustrated in Figure 1, the loss $\phi(v)$ is convex, monotonically decreasing, linear (with slope $-1$) for large negative $v$, constant for large positive $v$, and has slope $-.5$ at the origin. The set of losses compatible with an IS link is, thus, strongly constrained. The only degrees of freedom are in the behavior of the function around the origin. This is not surprising, since the only degrees of freedom of the sigmoid itself are in its behavior within this region.

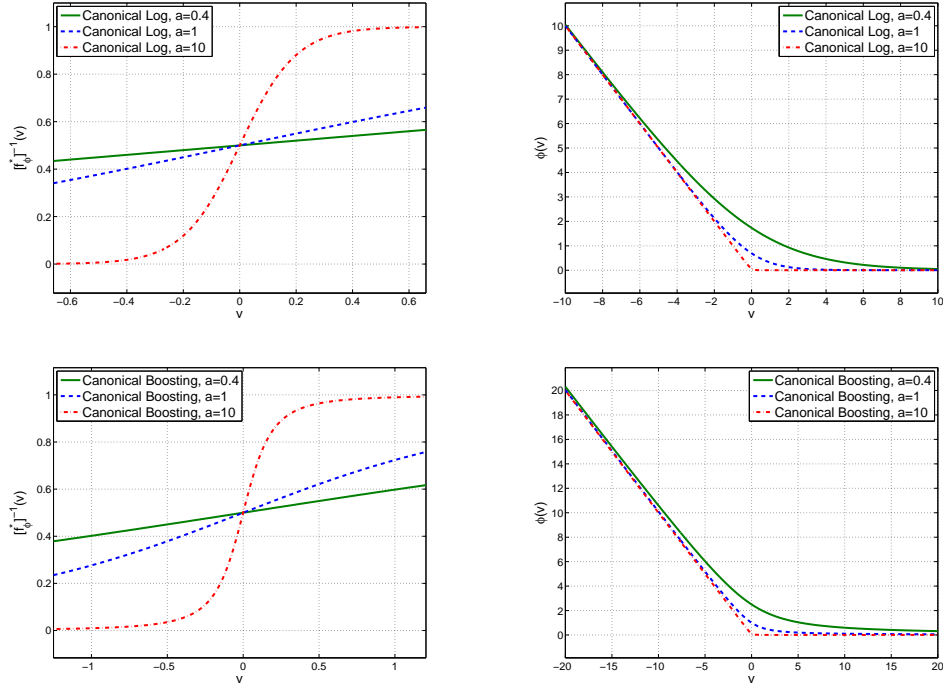

Figure 2: canonical link (left) and loss (right) for various values of $a$. (Top) logistic, (bottom) boosting.

What is interesting is that these are the degrees of freedom that control the margin characteristics of the loss $\phi$. Hence, by controlling the behavior of the IS link around the origin, it is possible to control the margin of the optimal classifier. In particular, the margin is a decreasing function of the curvature of the loss at the origin, $\phi^{(2)}(0)$. Since, from (18), $\phi^{(2)}(0) = ([f_\phi^*]^{-1})^{(1)}(0)$, the margin can be controlled by varying the slope of $[f_\phi^*]^{-1}$ at the origin.

## 5 Variable margin loss functions

The results above enable the derivation of families of canonical losses with controllable margin. In Section 3, we have seen that the boosting loss is not canonical, but there is a canonical loss for the minimum risk of boosting. We consider a parametric extension of this risk,

$$J(\eta; a) = \frac{-2}{a}\sqrt{\eta(1-\eta)}, \quad a > 0. \tag{25}$$

From (16), the canonical optimal link is

$$f_\phi^*(\eta; a) = \frac{2\eta - 1}{a\sqrt{\eta(1-\eta)}} \tag{26}$$

and it can be shown that

$$[f_\phi^*]^{-1}(v; a) = \frac{1}{2} + \frac{av}{2\sqrt{4 + (av)^2}} \tag{27}$$

is an IS link, i.e. satisfies (19)-(24). Using (18), the corresponding canonical loss is

$$\phi(v; a) = \frac{1}{2a}(\sqrt{4 + (av)^2} - av). \tag{28}$$

Because it shares the minimum risk of boosting, we refer to this loss as the *canonical boosting loss*. It is plotted in Figure 2, along with the inverse link, for various values of $a$. Note that the inverse

Table 2: Margin parameter value $a$ of rank 1 for each of the ten UCI datasets.

| UCI dataset# | #1 | #2 | #3 | #4 | #5 | #6 | #7 | #8 | #9 | #10 |
|---|---|---|---|---|---|---|---|---|---|---|
| Canonical Log | 0.4 | 0.5 | 0.6 | 0.3 | 0.1 | 2 | 0.5 | 0.1 | 0.2 | 0.2 |
| Canonical Boost | 0.9 | 6 | 2 | 2 | 0.4 | 3 | 0.2 | 4 | 0.2 | 0.9 |

link is indeed sigmoidal, and that the margin is determined by $a$. Since $\phi^{(2)}(0; a) = \frac{a}{4}$, the margin increases with decreasing $a$.

It is also possible to derive variable margin extensions of existing canonical losses. For example, consider the parametric extension of the minimum risk of logistic regression

$$J(\eta; a) = \frac{1}{a}\eta \log(\eta) + \frac{1}{a}(1 - \eta) \log(1 - \eta). \tag{29}$$

From (16),

$$[f_\phi^*](v; a) = \frac{1}{a} \log \frac{\eta}{1 - \eta} \quad [f_\phi^*]^{-1}(v; a) = \frac{e^{av}}{1 + e^{av}}. \tag{30}$$

This is again a sigmoidal inverse link and, from (18),

$$\phi(v; a) = \frac{1}{a} \left[\log(1 + e^{av}) - av\right]. \tag{31}$$

We denote this loss the *canonical logistic loss*. It is plotted in Figure 2, along with the corresponding inverse link for various $a$. Since $\phi^{(2)}(0; a) = \frac{a}{4}$, the margin again increases with decreasing $a$.

Note that, in (28) and (31), margin control is not achieved by simply rescaling the domain of the loss function. e.g. just replacing $\log(1 + e^{-v})$ by $\log(1 + e^{-av})$ in the case of logistic regression. This would have no impact in classification accuracy, since it would just amount to a change of scale of the original feature space. While this type of re-scaling occurs in both families of loss functions above (which are both functions of $av$), it is localized around the origin, and only influences the margin properties of the loss. As can be seen seen in Figure 2 all loss functions are identical away from the origin. Hence, varying $a$ is conceptually similar to varying the bandwidth of an SVM kernel. This suggests that the margin parameter $a$ could be cross-validated to achieve best performance.

## 6 Experiments

A number of easily reproducible experiments were conducted to study the effect of variable margin losses on the accuracy of the resulting classifiers. Ten binary UCI data sets were considered: (#1)sonar, (#2)breast cancer prognostic, (#3)breast cancer diagnostic, (#4)original Wisconsin breast cancer, (#5)Cleveland heart disease, (#6)tic-tac-toe, (#7)echo-cardiogram, (#8)Haberman's survival (#9)Pima-diabetes and (#10)liver disorder. The data was split into five folds, four used for training and one for testing. This produced five training-test pairs per dataset. The GradientBoost algorithm [8], with histogram-based weak learners, was then used to design boosted classifiers which minimize the canonical logistic and boosting losses, for various margin parameters. GradientBoost was adopted because it can be easily combined with the different losses, guaranteeing that, other than the loss, every aspect of classifier design is constant. This makes the comparison as fair as possible. 50 boosting iterations were applied to each training set, for 19 values of $a \in \{0.1, 0.2, ..., 0.9, 1, 2, ..., 10\}$. The classification accuracy was then computed per dataset, by averaging over its five train/test pairs.

Since existing algorithms can be seen as derived from special cases of the proposed losses, with $a = 1$, it is natural to inquire whether other values of the margin parameter will achieve best performance. To explore this question we show, in Figure-1, the average rank of the classifier designed with each loss and margin parameter $a$. To produce the plot, a classifier was trained on each dataset, for all 19 values of $a$. The results were then ranked, with rank $1$ ($19$) being assigned to the $a$ parameter of smallest (largest) error. The ranks achieved with each $a$ were then averaged over the ten datasets, as suggested in [9]. For the canonical logistic loss, the best values of $a$ is in the range $0.2 \leq a \leq 0.3$. Note that the average rank for this range (between $5$ and $6$), is better than that (close to $7$) achieved with the logistic loss of LogitBoost [2] ($a = 1$). In fact, as can be seen from Table 2, the canonical

Table 3: Classification error for each loss function and UCI dataset.

| UCI dataset# | #1 | #2 | #3 | #4 | #5 | #6 | #7 | #8 | #9 | #10 |
|---|---|---|---|---|---|---|---|---|---|---|
| Canonical Log | **11.2** | **11.4** | **8** | **5.6** | **12.4** | 11.8 | 7 | **18.8** | 38.2 | **27** |
| LogitBoost ($a = 1$) | 11.6 | 12.4 | 10 | 6.6 | 13.4 | 48.6 | 6.8 | 21.2 | 39.6 | 28.4 |
| Canonical Boost | 12.6 | 11.6 | 21 | 18.6 | 17.6 | **7.2** | 6 | 21.8 | **37.6** | 28.6 |
| Canonical Boost, $a = 1$ | 13.2 | 12.4 | 21 | 18.6 | 18.6 | 50.8 | 7.2 | 21.2 | 39.4 | 28.2 |
| AdaBoost | 11.4 | **11.4** | 9.4 | 6.4 | 14 | 28 | 6.6 | 21.8 | 41.2 | 28.2 |

Table 4: Classification error for each loss function and UCI dataset.

| UCI dataset# | #1 | #2 | #3 | #4 | #5 | #6 | #7 | #8 | #9 | #10 |
|---|---|---|---|---|---|---|---|---|---|---|
| Canonical Log, $a = 0.2$ | 13.2 | **15** | 8.4 | **5** | **11.2** | 56.2 | **6.8** | 24 | 39.8 | **25.8** |
| Canonical Boost, $a = 0.2$ | 12.6 | 14.8 | 17.2 | 18.6 | 12 | 56.8 | **6.8** | **23.2** | **38.4** | 26.4 |
| LogitBoost ($a = 1$) | 12.4 | 15.4 | 8.6 | 5.6 | 11.4 | 46 | 7.2 | 25 | 40.4 | 26.4 |
| AdaBoost | **11.4** | 15.2 | 9.2 | 6 | 11.4 | **21.6** | 7.4 | **23.2** | 42.8 | 26.6 |

logistic loss with $a = 1$ did not achieve rank 1 on any dataset, whereas canonical logistic losses with $0.2 \leq a \leq 0.3$ were top ranked on 3 datasets (and with $0.1 \leq a \leq 0.4$ on 6). For the canonical boosting loss, there is also a range ($0.8 \leq a \leq 2$) that produces best results. We note that the $a$ values of the two losses are not directly comparable. This can be seen from Figure-2 where $a = 0.4$ produces a loss of much larger margin for canonical boosting. Furthermore, the canonical boosting loss has a heavier tail and approaches zero more slowly than the canonical logistic loss.

Although certain ranges of margin parameters seem to produce best results for both canonical loss functions, the optimal parameter value is likely to be dataset dependent. This is confirmed by Table 2 which presents the parameter value of rank 1 for each of the ten datasets. Improved performance should thus be possible by cross-validating the margin parameter $a$. Table 3 presents the 5-fold cross validation test error (# of misclassified points) obtained for each UCI dataset and canonical loss. The table also shows the results of AdaBoost, LogitBoost (canonical logistic, $a = 1$), and canonical boosting loss with $a = 1$. Cross validating the margin results in better performance for 9 out of 10 (8 out 10) datasets for the canonical logistic (boosting) loss, when compared to the fixed margin ($a = 1$) counterparts. When compared to the existing algorithms, at least one of the margin-tunned classifiers is better than both Logit and AdaBoost for each dataset.

Under certain experimental conditions, cross validation might not be possible or computationally feasible. Even in this case, it may be better to use a value of $a$ other than the standard $a = 1$. Table-4 presents results for the case where the margin parameter is fixed at $a = 0.2$ for both canonical loss functions. In this case, canonical logistic and canonical boosting outperform *both* LogitBoost and AdaBoost in 7 and 5 of the ten datasets, respectively. The converse, i.e. LogitBoost and AdaBoost outperforming both canonical losses only happens in 2 and 3 datasets, respectively.

# 7 Conclusion

The probability elicitation approach to loss function design, introduced in [3], enables the derivation of new Bayes consistent loss functions. Yet, because the procedure is not fully constructive, this requires trial and error. In general, it is difficult to anticipate the properties, and shape, of a loss function that results from combining a certain minimal risk with a certain link function. In this work, we have addressed this problem for the class of canonical risks. We have shown that the associated canonical loss functions lend themselves to analysis, due to a simple connection between the associated minimum conditional risk and optimal link functions. This analysis was shown to enable a precise characterization of 1) the relationships between loss, optimal link, and minimum risk, and 2) the properties of the loss whenever the optimal link is in the family of inverse sigmoid functions. These properties were then exploited to design parametric families of loss functions with explicit margin control. Experiments with boosting algorithms derived from these variable margin losses have shown better performance than those of classical algorithms, such as AdaBoost or LogitBoost.

# References

[1] V. N. Vapnik, *Statistical Learning Theory*.   John Wiley Sons Inc, 1998.

[2] J. Friedman, T. Hastie, and R. Tibshirani, "Additive logistic regression: A statistical view of boosting," *Annals of Statistics*, 2000.

[3] H. Masnadi-Shirazi and N. Vasconcelos, "On the design of loss functions for classification: theory, robustness to outliers, and savageboost," in *NIPS*, 2008, pp. 1049–1056.

[4] L. J. Savage, "The elicitation of personal probabilities and expectations," *Journal of the American Statistical Association*, vol. 66, pp. 783–801, 1971.

[5] C. Leistner, A. Saffari, P. M. Roth, and H. Bischof, "On robustness of on-line boosting - a competitive study," in *IEEE ICCV Workshop on On-line Computer Vision*, 2009.

[6] A. Buja, W. Stuetzle, and Y. Shen, "Loss functions for binary class probability estimation and classification: Structure and applications," 2006.

[7] T. Zhang, "Statistical behavior and consistency of classification methods based on convex risk minimization," *Annals of Statistics*, 2004.

[8] J. H. Friedman, "Greedy function approximation: A gradient boosting machine," *The Annals of Statistics*, vol. 29, no. 5, pp. 1189–1232, 2001.

[9] J. Demšar, "Statistical comparisons of classifiers over multiple data sets," *The Journal of Machine Learning Research*, vol. 7, pp. 1–30, 2006.

